# Memory-based Reinforcement Learning: Efficient Computation with Prioritized Sweeping

**Andrew W. Moore**
awm@ai.mit.edu
NE43-759 MIT AI Lab.
545 Technology Square
Cambridge MA 02139

**Christopher G. Atkeson**
cga@ai.mit.edu
NE43-771 MIT AI Lab.
545 Technology Square
Cambridge MA 02139

## Abstract

We present a new algorithm, Prioritized Sweeping, for efficient prediction and control of stochastic Markov systems. Incremental learning methods such as Temporal Differencing and Q-learning have fast real time performance. Classical methods are slower, but more accurate, because they make full use of the observations. Prioritized Sweeping aims for the best of both worlds. It uses all previous experiences both to prioritize important dynamic programming sweeps and to guide the exploration of state-space. We compare Prioritized Sweeping with other reinforcement learning schemes for a number of different stochastic optimal control problems. It successfully solves large state-space real time problems with which other methods have difficulty.

## 1 STOCHASTIC PREDICTION

The paper introduces a memory-based technique, *prioritized sweeping*, which is used both for stochastic prediction and reinforcement learning. A fuller version of this paper is in preparation [Moore and Atkeson, 1992]. Consider the 500 state Markov system depicted in Figure 1. The system has sixteen absorbing states, depicted by white and black circles. The prediction problem is to estimate, for every state, the long-term probability that it will terminate in a white, rather than black, circle. The data available to the learner is a sequence of observed state transitions. Let us consider two existing methods along with prioritized sweeping.

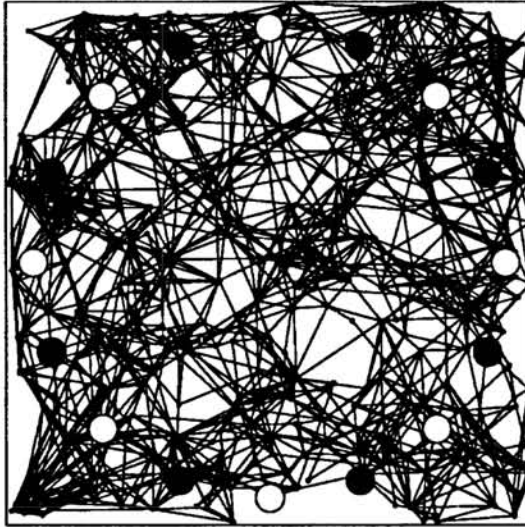

Figure 1: A 500-state Markov system. Each state has a random number (mean 5) of random successors chosen within the local neighborhood.

**Temporal Differencing (TD)** is an elegant incremental algorithm [Sutton, 1988] which has recently had success with a very large problem [Tesauro, 1991].

The **classical method** proceeds by building a maximum likelihood model of the state transitions. $q_{ij}$ (the transition probability from $i$ to $j$) is estimated by

$$\hat{q}_{ij} = \frac{\text{Number of observations } i \to j}{\text{Number of occasions in state } i} \qquad (1)$$

After $t + 1$ observations the new absorption probability estimates are computed to satisfy, for each terminal state $k$, the linear system

$$\hat{\pi}_{ik}[t+1] = \hat{q}_{ik} + \sum_{j \in \text{succs}(i) \cap \textbf{NONTERMS}} \hat{q}_{ij}\hat{\pi}_{jk}[t+1] \qquad (2)$$

where the $\hat{\pi}_{ik}[t]$'s are the absorption probabilities we are trying to learn, where $\text{succs}(i)$ is the set of all states which have been observed as immediate successors of $i$ and **NONTERMS** is the set of non-terminal states.

This set of equations is solved after each transition is observed. It is solved using Gauss-Seidel—an iterative method. What initial estimates should be used to start the iteration? An excellent answer is to use the previous absorption probability estimates $\hat{\pi}_{ik}[t]$.

**Prioritized sweeping** is designed to combine the advantages of the classical method with the advantages of TD. It is described in the next section, but let us first examine performance on the original 500-state example of Figure 1. Figure 2 shows the result. TD certainly learns: by 100,000 observations it is estimating the terminal-white probability to an RMS accuracy of 0.1. However, the performance of the classical method appears considerably better than TD: the same error of 0.1 is obtained after only 3000 observations.

Figure 3 indicates why temporal differencing may nevertheless often be more useful. TD requires far less computation per observation, and so can obtain more data in real time. Thus, after 300 seconds, TD has had 250,000 observations and is down

| Mean ± Standard Dev'n | TD | Classical | Pri. Sweep |
|---|---|---|---|
| After 100,000 observations | $0.40 \pm 0.077$ | $0.024 \pm 0.0063$ | $0.024 \pm 0.0061$ |
| After 300 seconds | $0.079 \pm 0.067$ | $0.23 \pm 0.038$ | $0.021 \pm 0.0080$ |

Table 1: RMS prediction error: mean and standard deviation for ten experiments.

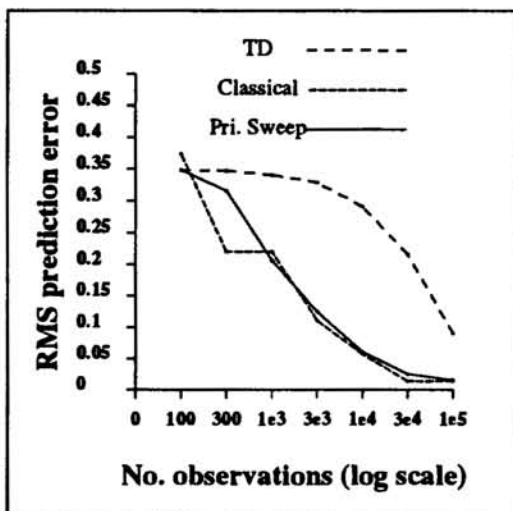

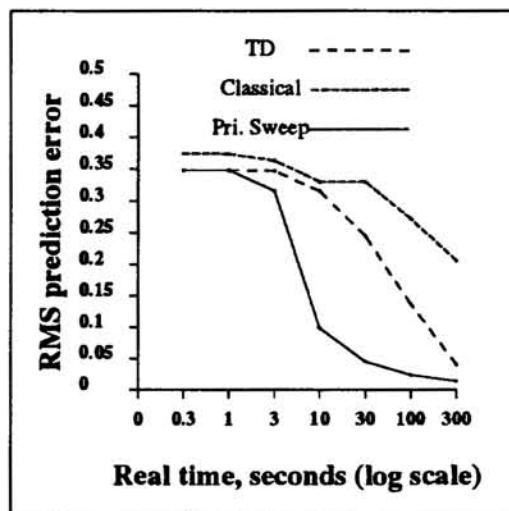

Figure 2: RMS prediction against observation during three learning algorithms.

Figure 3: RMS prediction against real time

to an error of 0.05, whereas even after 300 seconds the classical method has only 1000 observations and a much cruder estimate.

In the same figures we see the motivation behind prioritized sweeping. Its performance relative to observations is almost as good as the classical method, while its performance relative to real time is even better than TD.

The graphs in Figures 2 and 3 were based on only one learning experiment each. Ten further experiments, each with a different random 500 state problem, were run. The results are given in Table 1.

## 2 PRIORITIZED SWEEPING

A longer paper [Moore and Atkeson, 1992] will describe the algorithm in detail. Here we summarize the essential insights, and then simply present the algorithm in Figure 4. The closest relation to prioritized sweeping is the search scheduling technique of the $A^\star$ algorithm [Nilsson, 1971]. Closely related research is being performed by [Peng and Williams, 1992] into a similar algorithm to prioritized sweeping, which they call Dyna-Q-queue.

- The memory requirements of learning a $N_s \times N_s$ matrix, where $N_s$ is the number of states, may initially appear prohibitive, especially since we intend to operate with more than 10,000 states. However, we need only allocate memory for the

1. Promote state $i_{\text{recent}}$ (the source of the most recent transition) to top of priority queue.

2. While we are allowed further processing and priority queue not empty

   2.1 Remove the top state from the priority queue. Call it $i$

   2.2 $\Delta_{\text{max}} = 0$

   2.3 for each $k \in$ **TERMS**

$$\rho_{\text{new}} = \hat{q}_{ik} + \sum_{j \in \text{succs}(i) \cap \textbf{NONTERMS}} \hat{q}_{ij} \hat{\pi}_{jk}$$

$$\Delta := |\rho_{\text{new}} - \hat{\pi}_{ik}|$$

$$\hat{\pi}_{ik} := \rho_{\text{new}}$$

$$\Delta_{\text{max}} := \max(\Delta_{\text{max}}, \Delta)$$

   2.4 for each $i' \in \text{preds}(i)$

$$P := \hat{q}_{i'i} \Delta_{\text{max}}$$

   if $i'$ not on queue, or $P$ exceeds the current priority of $i'$, then promote $i'$ to new priority $P$.

Figure 4: The prioritized sweeping algorithm. This sequence of operations is executed each time a transition is observed.

experiences the system actually has, and for a wide class of physical systems there is not enough time in the lifetime of the physical system to run out of memory.

- We keep a record of all predecessors of each state. When the eventual absorption probabilities of a state are updated, its predecessors are alerted that they may need to change. A priority value is assigned to each predecessor according to how large this change could be possibly be, and it is placed in a priority queue.

- After each real-world observation $i \rightarrow j$, the transition probability estimate $\hat{q}_{ij}$ is updated along with the probabilities of transition to all other previously observed successors of $i$. Then state $i$ is promoted to the top of the priority queue so that its absorption probabilities are updated immediately. Next, we continue to process further states from the top of the queue. Each state that is processed may result in the addition or promotion of its predecessors within the queue. This loop continues for a preset number of processing steps or until the queue empties.

If a real world observation is interesting, all its predecessors and their earlier ancestors quickly find themselves near the top of the priority queue. On the other hand, if the real world observation is unsurprising, then the processing immediately proceeds to other, more important areas of state-space which had been under consideration on the previous time step. These other areas may be different from those in which the system currently finds itself.

| | 15 States | 117 States | 605 States | 4528 States |
|---|---|---|---|---|
| | 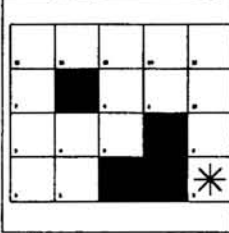 | 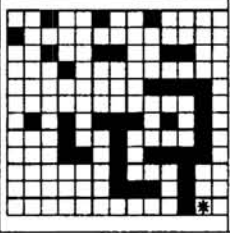 | 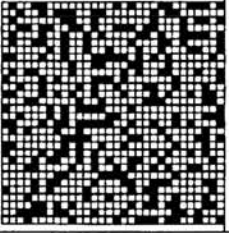 | 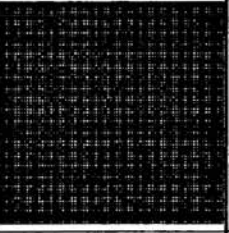 |
| Q | 800 | > 25000 | > 500000 | > 500000 |
| Dyna-PI+ | 400 | 500 | 36000 | > 500000 |
| Dyna-OPT | 300 | 900 | 21000 | 245000 |
| PriSweep | 150 | 1200 | 6000 | 59000 |

Table 2: Number of observations before 98% of decisions were subsequently optimal. Dyna and Prioritized Sweeping were each allowed to process ten states per real-world observation.

## 3   LEARNING CONTROL FROM REINFORCEMENT

Prioritized sweeping is also directly applicable to stochastic control problems. Remembering all previous transitions allows an additional advantage for control—exploration can be guided towards areas of state space in which we predict we are ignorant. This is achieved using the exploration philosophy of [Kaelbling, 1990] and [Sutton, 1990]: optimism in the face of uncertainty.

## 4   RESULTS

Results of some maze problems of significant size are shown in Table 2. Each state has four actions: one for each direction. Blocked actions do not move. One goal state (the star in subsequent figures) gives 100 units of reward, all others give no reward, and there is a discount factor of 0.99. Trials start in the bottom left corner. The system is reset to the start state whenever the goal state has been visited ten times since the last reset. The reset is outside the learning task: it is not observed as a state transition. Prioritized sweeping is tested against a highly tuned Q-learner [Watkins, 1989] and a highly tuned Dyna [Sutton, 1990]. The optimistic experimentation method (described in the full paper) can be applied to other algorithms, and so the results of optimistic Dyna-learning is also included.

The same mazes were also run as a stochastic problem in which requested actions were randomly corrupted 50% of the time. The gap between Dyna-OPT and Prioritized Sweeping was reduced in these cases. For example, on a stochastic 4528-state maze Dyna-OPT took 310,000 steps and Prioritized sweeping took 200,000.

We also have results for a five state bench-mark problem described in [Sato et al., 1988, Barto and Singh, 1990]. Convergence time is reduced by a factor of twenty over the incremental methods.

|                      | Experiences to converge | Real time to converge |
| -------------------- | :---------------------: | :-------------------: |
| Q                    | never                   |                       |
| Dyna–PI+             | never                   |                       |
| Optimistic Dyna      | 55,000                  | 1500 secs             |
| Prioritized Sweeping | 14,000                  | 330 secs              |

Table 3: Performance on the deterministic rod-in-maze task. Both Dynas and prioritized sweeping were allowed 100 backups per experience.

Finally we consider a task with a 3-d state space quantized into 15,000 potential discrete states (not all reachable). The task is shown in Figure 5 and involves finding the shortest path for a rod which can be rotated and translated.

Q, Dyna–PI+, Optimistic Dyna and prioritized sweeping were all tested. The results are in Table 3. Q and Dyna–PI+ did not even travel a quarter of the way to the goal, let alone discover an optimal path, within 200,000 experiences. Optimistic Dyna and prioritized sweeping both eventually converged, with the latter requiring a third the experiences and a fifth the real time.

When 2000 backups per experience were permitted, instead of 100, then both optimistic Dyna and prioritized sweeping required fewer experiences to converge. Optimistic Dyna took 21,000 experiences instead of 55,000 but took 2,900 seconds—almost twice the real time. Prioritized sweeping took 13,500 instead of 14,000 experiences—very little improvement, but it used no extra time. This indicates that for prioritized sweeping, 100 backups per observation is sufficient to make almost complete use of its observations, so that all the long term reward $(J_i)$ estimates are very close to the estimates which would be globally consistent with the transition probability estimates $(\hat{q}_{ij}^{a})$. Thus, we conjecture that even full dynamic programming after each experience (which would take days of real time) would do little better.

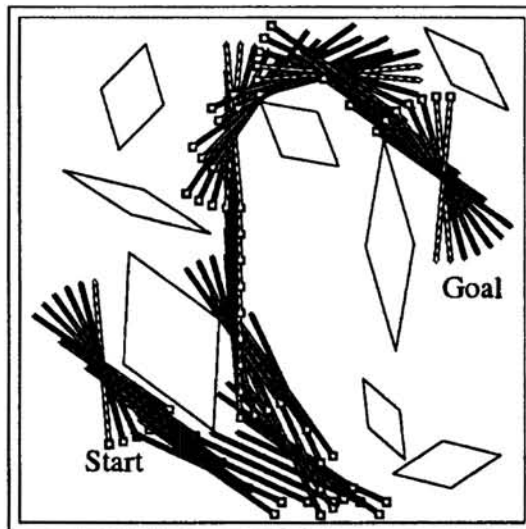

Figure 5: A three-DOF problem, and the optimal solution path.

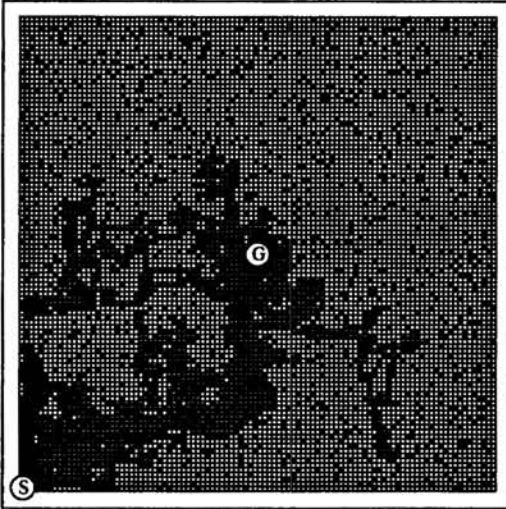

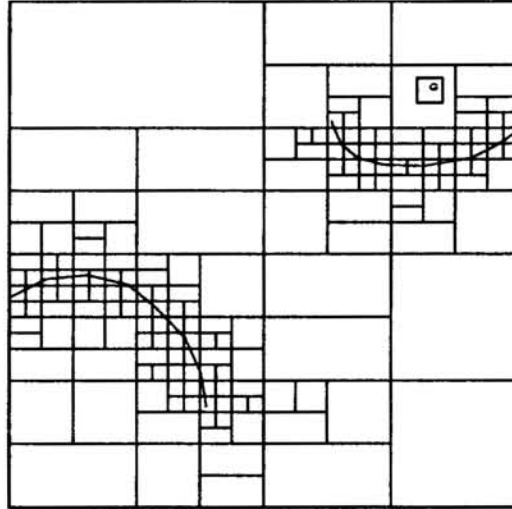

Figure 6: Dotted states are all those vis-
ited when the Manhattan heuristic was
used

Figure 7: A kd-tree tessellation of state
space of a sparse maze

## 5  DISCUSSION

Our investigation shows that Prioritized Sweeping can solve large state-space real-
time problems with which other methods have difficulty. An important extension
allows heuristics to constrain exploration decisions. For example, in finding an
optimal path through a maze, many states need not be considered at all. Figure 6
shows the areas explored using a Manhattan heuristic when finding the optimal
path from the lower left to the center. For some tasks we may be even satisfied to
cease exploration when we have obtained a solution known to be, say, within 50%
of the optimal solution. This can be achieved by using a heuristic which lies: it tells
us that the best possible reward-to-go is that of a path which is twice the length of
the true shortest possible path.

Furthermore, another promising avenue is prioritized sweeping in conjunction with
kd-tree tessellations of state space to concentrate prioritizing sweeping on the im-
portant regions [Moore, 1991]. Other benefits of the memory-based approach, de-
scribed in [Moore, 1992], allow us to control forgetting in changing environments
and automatic scaling of state variables.

## Acknowledgements

Thanks to Mary Soon Lee, Satinder Singh and Rich Sutton for useful comments
on an early draft. Andrew W. Moore is supported by a Postdoctoral Fellowship
from SERC/NATO. Support was also provided under Air Force Office of Scientific
Research grant AFOSR-89-0500, an Alfred P. Sloan Fellowship, the W. M. Keck
Foundation Associate Professorship in Biomedical Engineering, Siemens Corpora-
tion, and a National Science Foundation Presidential Young Investigator Award to
Christopher G. Atkeson.

# References

[Barto and Singh, 1990] A. G. Barto and S. P. Singh. On the Computational Economics of Reinforcement Learning. In D. S. Touretzky, editor, *Connectionist Models: Proceedings of the 1990 Summer School.* Morgan Kaufmann, 1990.

[Kaelbling, 1990] L. P. Kaelbling. Learning in Embedded Systems. PhD. Thesis; Technical Report No. TR-90-04, Stanford University, Department of Computer Science, June 1990.

[Moore and Atkeson, 1992] A. W. Moore and C. G. Atkeson. Memory-based Reinforcement Learning: Converging with Less Data and Less Real Time. In preparation, 1992.

[Moore, 1991] A. W. Moore. Variable Resolution Dynamic Programming: Efficiently Learning Action Maps in Multivariate Real-valued State-spaces. In L. Birnbaum and G. Collins, editors, *Machine Learning: Proceedings of the Eighth International Workshop.* Morgan Kaufman, June 1991.

[Moore, 1992] A. W. Moore. Fast, Robust Adaptive Control by Learning only Forward Models. In J. E. Moody, S. J. Hanson, and R. P. Lippman, editors, *Advances in Neural Information Processing Systems 4.* Morgan Kaufmann, April 1992.

[Nilsson, 1971] N. J. Nilsson. *Problem-solving Methods in Artificial Intelligence.* McGraw Hill, 1971.

[Peng and Williams, 1992] J. Peng and R. J. Williams. Efficient Search Control in Dyna. College of Computer Science, Northeastern University, March 1992.

[Sato et al., 1988] M. Sato, K. Abe, and H. Takeda. Learning Control of Finite Markov Chains with an Explicit Trade-off Between Estimation and Control. *IEEE Trans. on Systems, Man, and Cybernetics*, 18(5):667–684, 1988.

[Sutton, 1988] R. S. Sutton. Learning to Predict by the Methods of Temporal Differences. *Machine Learning*, 3:9–44, 1988.

[Sutton, 1990] R. S. Sutton. Integrated Architecture for Learning, Planning, and Reacting Based on Approximating Dynamic Programming. In *Proceedings of the 7th International Conference on Machine Learning.* Morgan Kaufman, June 1990.

[Tesauro, 1991] G. J. Tesauro. Practical Issues in Temporal Difference Learning. RC 17223 (76307), IBM T. J. Watson Research Center, NY, 1991.

[Watkins, 1989] C. J. C. H. Watkins. Learning from Delayed Rewards. PhD. Thesis, King's College, University of Cambridge, May 1989.
